# Bandit Algorithms boost motor-task selection for Brain Computer Interfaces

**Joan Fruitet**
INRIA, Sophia Antipolis
2004 Route des Lucioles
06560 Sophia Antipolis, France
joan.fruitet@inria.fr

**Alexandra Carpentier**
Statistical Laboratory, CMS
Wilberforce Road, Cambridge
CB3 0WB UK
a.carpentier@statslab.cam.ac.uk

**Rémi Munos**
INRIA Lille - Nord Europe
40, avenue Halley
59000 Villeneuve d'ascq, France
remi.munos@inria.fr

**Maureen Clerc**
INRIA, Sophia Antipolis
2004 Route des Lucioles
06560 Sophia Antipolis, France
Maureen.Clerc@inria.fr

## Abstract

Brain-computer interfaces (BCI) allow users to "communicate" with a computer without using their muscles. BCI based on sensori-motor rhythms use imaginary motor tasks, such as moving the right or left hand, to send control signals. The performances of a BCI can vary greatly across users but also depend on the tasks used, making the problem of appropriate task selection an important issue. This study presents a new procedure to automatically select as fast as possible a discriminant motor task for a brain-controlled button. We develop for this purpose an adaptive algorithm, *UCB-classif*, based on the stochastic bandit theory. This shortens the training stage, thereby allowing the exploration of a greater variety of tasks. By not wasting time on inefficient tasks, and focusing on the most promising ones, this algorithm results in a faster task selection and a more efficient use of the BCI training session. Comparing the proposed method to the standard practice in task selection, for a fixed time budget, *UCB-classif* leads to an improved classification rate, and for a fixed classification rate, to a reduction of the time spent in training by $50\%$.

## 1 Introduction

Scalp recorded electroencephalography (EEG) can be used for non-muscular control and communication systems, commonly called brain-computer interfaces (BCI). BCI allow users to "communicate" with a computer without using their muscles. The communication is made directly through the electrical activity from the brain, collected by EEG in real time. This is a particularly interesting prospect for severely handicapped people, but it can also be of use in other circumstances, for instance for enhanced video games.

A possible way of communicating through the BCI is by using sensori-motor rhythms (SMR), which are modulated in the course of movement execution or movement imagination. The SMR corresponding to movement imagination can be detected after pre-processing the EEG, which is corrupted by important noise, and after training (see [1, 2, 3]). A well-trained classifier can then use features of the SMR in order to discriminate periods of imagined movement from resting periods, when the user is idle. The detected mental states can be used as buttons in a Brain Computer Interface, mimicking traditional interfaces such as keyboard or mouse button.

This paper deals with training a BCI corresponding to a single brain-controlled button (see [2, 4]), in which a button is pressed (and instantaneously released) when a certain imagined movement is detected. The important steps are thus to find a suitable imaginary motor task, and to train a

classifier. This is far from trivial, because appropriate tasks which can be well classified from the background resting state are highly variable among subjects; moreover, the classifier requires to be trained on a large set of labeled data. The setting up of such a brain-controlled button can be very time consuming, given that many training examples need to be acquired for each of the imaginary motor task to be tested.

The usual training protocol for a brain-controlled button is to display sequentially to the user a set of images, that serve as prompts to perform the corresponding imaginary movements. The collected data are used to train the classifier, and to select the imaginary movement that seems to provide the highest classification rate (compared to the background resting state). We refer to this imaginary movement as the "best imaginary movement". In this paper, we focus on the part of the training phase that consists in efficiently finding this best imaginary movement. This is an important problem, since the SMR collected by the EEG are heterogeneously noisy: some imaginary motor tasks will provide higher classification rates than others. In the literature, finding such imaginary motor tasks is deemed an essential issue (see [5, 6, 7]), but, to the best of our knowledge, no automatized protocol has yet been proposed to deal with it. We believe that enhancing the efficiency of the training phase is made even more essential by the facts that (i) the best imaginary movement differs from one user to another, e.g. the best imaginary movement for one user could be to imagine moving the right hand, and for the next, to imagine moving both feet (see [8]) and (ii) using a BCI requires much concentration, and a long training phase exhausts the user.

If an "oracle" were able to state what the best imaginary movement is, then the training phase would consist only in requiring the user to perform this imaginary movement. The training set for the classifier on this imaginary movement would be large, and no training time would be wasted in asking the user to perform sub-optimal and thus useless imaginary movements. The best imaginary movement is however not known in advance, and so the commonly used strategy (which we will refer to as *uniform*) consists in asking the user to perform all the movements a fixed number of times. An alternative strategy is to learn *while building the training set* what imaginary movements seem the most promising, and ask the classifier to perform these more often. This problem is quite archetypal to a field of Machine Learning called Bandit Theory (initiated in [9]). Indeed, the main idea in Bandit Theory is to mix the **Exploration** of the possible actions[1], and their **Exploitation** to *perform* the empirical best action.

**Contributions**　This paper builds on ideas of Bandit Theory, in order to propose an efficient method to select the best imaginary movement for the activation of a brain-controlled button. To the best of our knowledge, this is the first contribution to the automation and optimization of this task selection.

- We design a BCI experiment for imaginary motor task selection, and collect data on several subjects, for different imaginary motor tasks, in the aim of testing our methods.

- We provide a bandit algorithm (which is strongly inspired by the Upper Confidence Bound Algorithm of [10]) adapted to this classification problem. In addition, we propose several variants of this algorithm that are intended to deal with other slightly different scenarios that the practitioner might face. We believe that this bandit-based classification technique is of independent interest and could be applied to other task selection procedures under constraints on the samples.

- We provide empirical evidence that using such an algorithm considerably speeds up the training phase for the BCI. We gain up to $18\%$ in terms of classification rate, and up to $50\%$ in training time, when compared to the uniform strategy traditionally used in the literature.

The rest of the paper is organized as follows: in Section 2, we describe the EEG experiment we built in order to acquire data and simulate the training of a brain-controlled button. In Section 3, we model the task selection as a bandit problem, which is solved using an Upper Confidence Bound algorithm. We motivate the choice of this algorithm by providing a performance analysis. Section 4, which is the main focus of this paper, presents results on simulated experiments, and proves empirically the gain brought forth by adaptive algorithms in this setting. We then conclude this paper with further perspectives.

## 2 Material and protocol

BCI systems based on SMR rely on the users' ability to *control* their SMR in the mu (8-13Hz) and/or beta (16-24Hz) frequency bands [1, 2, 3]. Indeed, these rhythms are naturally modulated during real and imagined motor action.

More precisely, real and imagined movements similarly activate neural structures located in the sensori-motor cortex, which can be detected in EEG recordings through increases in power (event related synchronization or ERS) and/or decreases in power (event related de-synchronization or ERD) in the mu and beta frequency bands [11, 12]. Because of the homuncular organization of the sensori-motor cortex [13], different limb movements may be distinguished according to the spatial layout of the ERD/ERS.

BCI based on the control of SMR generally use movements lasting several seconds, that enable continuous control of multidimensional interfaces [1]. On the contrary this work targets a brain-controlled button that can be rapidly triggered by a short motor task [2, 4]. A vast variety of motor tasks can be used in this context, like imagining rapidly moving the hand, grasping an object, or kicking an imaginary ball. We remind that the best imaginary movement differs from one user to another (see [8]).

As explained in the Introduction, the use of a BCI must always be preceded by a training phase. In the case of a BCI managing a brain-controlled button through SMR, this training phase consists in displaying to the user a sequence of images corresponding to movements, that he/she must imagine performing. By processing the EEG, the SMR associated to the imaginary movements and to idle periods can be extracted. Collecting these labeled data results in a training set, which serves to train the classifier between the movements, and the idle periods. The imaginary movement with highest classification rate is then selected to activate the button in the actual use of the BCI.

The rest of this Section explains in more detail the BCI material and protocol used to acquire the EEG, and to extract the features from the signal.

### 2.1 The EEG experiment

The EEG experiment was similar to the training of a brain-controlled button: we presented, at random timing, cue images during which the subjects were asked to perform 2 second long motor tasks (intended to activate the button).

Six right-handed subjects, aged 24 to 39, with no disabilities, were sitting at 1.5m of a 23' LCD screen. EEG was recorded dat a sampling rate of 512Hz via 11 scalp electrodes of a 64-channel cap and amplified with a TMSI amplifier (see Figure 1). The OpenViBE platform [14] was used to run the experiment. The signal was filtered in time through a band-pass filter, and in space through a surface Laplacian to increase the signal to noise ratio.

The experiment was composed of 5 to 12 blocks of approximately 5 minutes. During each block, 4 cue images were presented for 2 seconds in a random order, 10 times each. The time between two image presentations varied between 1.5s and 10s. Each cue image was a prompt for the subject to perform or imagine the corresponding motor action during 2 seconds, namely moving the right or left hand, the feet or the tongue.

### 2.2 Feature extraction

In the case of short motor tasks, the movement (real or imagined) produces an ERD in the mu and beta bands during the task, and is followed by a strong ERS [4] (sometimes called beta rebound as it is most easily seen in the beta frequency band).

We extracted features of the mu and beta bands during the 2-second windows of the motor action and in the subsequent 1.5 seconds of signal in order to use the bursts of mu and beta power (ERS or rebound) that follow the indicated movement. Figure 1 shows a time-frequency map on which the movement and rebound windows are indicated. One may observe that, during the movement, the power in the mu and beta bands decreases (ERD) and that, approximately 1 second after the movement, it increases to reach a higher level than in the resting state (ERS).

More precisely, the features were chosen as the power around 12Hz and 18Hz extracted at 3 electrodes over the sensori-motor cortex (C3, C4 and Cz). Thus, 6 features are extracted during the movement and 6 during the rebound. The lengths and positions of the windows and the frequency bands were chosen according to a preliminary study with one of the subjects and were deliberately kept fixed for the other subjects.

One of the goals of our algorithm is to be able to select the best task among a large number of tasks. However, in our experiment, only a limited number of tasks were used (four), because we limited the length of the sessions in order not to tire the subjects. To demonstrate the usefulness of our method for a larger number of tasks, we decided to create artificial (degraded) tasks by mixing the features of one of the real tasks (the feet) with different proportions of the features extracted during the resting period.

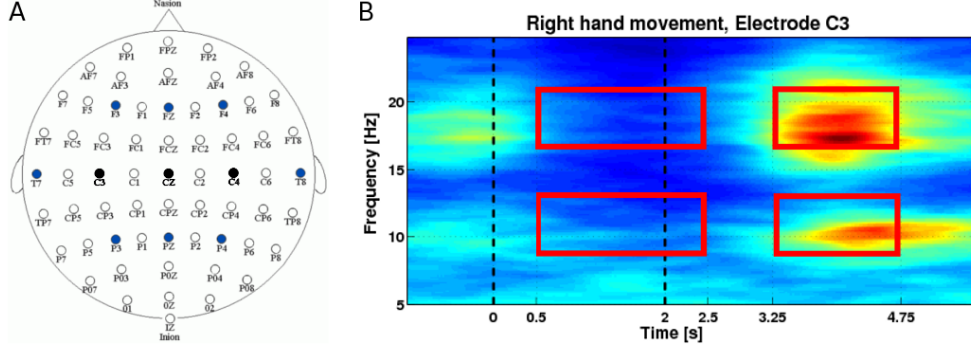

Figure 1: A: Layout of the 64 EEG cap, with (in black) the 3 electrodes from which the features are extracted. The electrodes marked in blue/grey are used for the Laplacian. B: Time-frequency map of the signal recorded on electrode C3, for a right hand movement lasting 2 seconds (subject 1). Four features (red windows) are extracted for each of the 3 electrodes.

## 2.3 Evaluation of performances

For each task $k$, we can classify between when the subject is inactive and when he/she is performing task $k$. Consider a sample $(X, Y) \sim \mathcal{D}_k$ where $\mathcal{D}_k$ is the distribution of the data restricted to task $k$ and the idle task (task 0), $X$ is the feature set, and $Y$ is the label (1 if the sample corresponds to task $k$ and 0 otherwise).

We consider a compact set of classifiers $\mathcal{H}$. Define the best classifier in $\mathcal{H}$ for task $k$ as $h_k^* = \arg\min_{h \in \mathcal{H}} \mathbb{E}_{(X,Y) \sim \mathcal{D}_k}[\mathbf{1}\{h(X) \neq Y\}]$. Define the theoretical loss $r_k^*$ of a task $k$ as the probability of labeling incorrectly a new data drawn from $\mathcal{D}_k$ with the best classifier $h_k^*$, that is to say $r_k^* = 1 - \mathbb{P}_{(X,Y) \sim \mathcal{D}}(h_k^*(X) \neq Y)$.

At time $t$, there are $T_{k,t} + T_{0,t}$ samples $(X_i, Y_i)_{i \leq T_{k,t}+T_{0,t}}$ (where $T_{k,t}$ is the number of samples for task $k$, and $T_{0,t}$ is the number of samples for the idle task) that are available. With these data, we build the empirical minimizer of the loss $\hat{h}_{k,t} = \arg\min_{h \in \mathcal{H}} \left[ \sum_{i=1}^{T_{k,t}+T_{0,t}} \mathbf{1}\{h(X_i) \neq Y_i\} \right]$. We define the empirical loss of this classifier $\hat{r}_{k,t} = 1 - \min_{h \in \mathcal{H}} \left[ \sum_{i=1}^{T_{k,t}+T_{0,t}} \mathbf{1}\{h(X_i) \neq Y_i\} \right]$.

Since during our experiments we collect, between each imaginary task, a sample of idle condition, we have $T_{0,t} \gg T_{k,t}$.

From Vapnik-Chervonenkis theory (see [15] and also the Supplementary Material), we obtain with probability $1 - \delta$, that the error in generalization of classifier $\hat{h}_{k,t}$ is not larger than $r_k^* + O\left(\sqrt{\frac{d \log(1/\delta)}{T_{k,n}}}\right)$, where $d$ is the VC dimension of the domain of $X$. This implies that the performance of the optimal empirical classifier for task $k$ is close to the performance of the optimal classifier for task $k$. Also with probability $1 - \delta$,

$$|\hat{r}_{k,t} - r_k^*| = O\left(\sqrt{\frac{d \log(1/\delta)}{T_{k,n}}}\right). \tag{1}$$

We consider in this paper linear classifiers. In this case, the VC dimension $d$ is the dimension of $X$, i.e. the number of features. The loss we considered ($(0, 1)$ loss) is difficult to minimize in practice because it is not convex. This is why we consider in this work the classifier $\hat{h}_{k,t}$ provided by linear SVM. We also estimate the performance $\hat{r}_{k,t}$ of this classifier by cross-validation: we use the leave-one-out technique when less than 8 samples of the task are available, and a 8-fold validation when more repetitions of the task have been recorded. As explained in [15], results similar to Equation 1 hold for this classifier.

We will use in the next Section the results of Equation 1, in order to select as fast as possible the task with highest $r_k^*$ and collect as many samples from it as possible.

# 3 A bandit algorithms for optimal task selection

In order to improve the efficiency of the training phase, it is important to find out as fast as possible what are the most promising imaginary tasks (i.e. tasks with large $r_k^*$). Indeed, it is important to collect as many samples as possible from the best imaginary movement, so that the classifier built for this task is as precise as possible.

In this Section, we propose the *UCB-Classif* algorithm, inspired by the Upper Confidence Bound algorithm in Bandit Theory (see [10]).

## 3.1 Modeling the problem by a multi-armed bandit

Let $K$ denote the number of different tasks[2] and $N$ the total number of rounds (the budget) of the training stage. Our goal is to find a presentation strategy for the images (i.e. that choose at each time-step $t \in \{1, \dots, N\}$ an image $k_t \in \{1, \dots, K\}$ to show), which allows to determine the "best", i.e. most discriminative imaginary movement, with highest classification rate in generalization). Note that, in order to learn an efficient classifier, we need as many training data as possible, so our presentation strategy should rapidly focus on the most promising tasks in order to obtain more samples from these rather than from the ones with small classification rate.

This issue is relatively close to the *stochastic bandit problem* [9]. The classical *stochastic bandit problem* is defined by a set of $K$ actions (pulling different arms of bandit machines) and to each action is assigned a reward distribution, initially unknown to the learner. At time $t \in \{1, \dots, N\}$, if we choose an action $k_t \in \{1, \dots, K\}$, we receive a reward sample drawn independently from the distribution of the corresponding action $k_t$. The goal is to find a sampling strategy which maximizes the sum of obtained rewards.

We model the $K$ different images to be displayed as the $K$ possible actions, and we define the reward as the classification rate of the corresponding motor action. In the *bandit problem*, pulling a bandit arm directly gives a stochastic reward which is used to estimate the distribution of this arm. In our case, when we display a new image, we obtain a new data sample for the selected imaginary movement, which provides one more data sample to train or test the corresponding classifier and thus obtain a more accurate performance. The main difference is that for the *stochastic bandit problem*, the goal is to maximize the sum of obtained rewards, whereas ours is to maximize the performance of the final classifier. However, the strategies are similar: since the distributions are initially unknown, one should first explore all the actions (exploration phase) but then rapidly select the best one (exploitation phase). This is called the *exploration-exploitation trade-off*.

## 3.2 The UCB-classif algorithm

The task presentation strategy is a close variant of the Upper Confidence Bound (UCB) algorithm of [10], which builds high probability Upper Confidence Bounds (UCB) on the mean reward value of each action, and selects at each time step the action with highest bound.

We adapt the idea of this UCB algorithm to our adaptive classification problem and call this algorithm *UCB-classif* (see the pseudo-code in Table 1). The algorithm builds a sequence of values $B_{k,t}$ defined as

$$B_{k,t} = \hat{r}_{k,t} + \sqrt{\frac{a \log N}{T_{k,t-1}}}, \tag{2}$$

where $\hat{r}_{k,t}$ represents an estimation of the classification rate built from a $q$-fold cross-validation technique and the $a$ corresponds to Equation 1 (see Supplementary Material for the precise theoretical value). The cross-validation uses a linear SVM classifier based on the $T_{k,t}$ data samples obtained (at time $t$) from task $k$. Writing $r_k^*$ the classification rate for the optimal linear SVM classifier (which would be obtained by using a infinite number of samples), we have the property that $B_{k,t}$ is a high probability upper bound on $r_k^*$: $\mathbb{P}(B_{k,t} < r_k^*)$ decreases to zero polynomially fast (with $N$).

The intuition behind the algorithm is that it selects at time $t$ an action $k_t$ either because it has a good classification rate $\hat{r}_{k,t}$ (thus it is interesting to obtain more samples from it, to perform exploitation) or because its classification rate is highly uncertain since it has not been sampled many times, i.e., $T_{k,t-1}$ is small and then $\sqrt{\frac{a \log N}{T_{k,t-1}}}$ is large (thus it is important to explore it more). With this strategy, the action that has the highest classification rate is presented more often. It is indeed important to

**The *UCB-Classif* Algorithm**
**Parameters:** $a$, $N$, $q$
Present each image $q = 3$ times (thus set $T_{k,qK} = q$).
**for** $t = qK + 1, \ldots, N$ **do**
    Evaluate the performance $\hat{r}_{k,t}$ of each action (by a 8-split Cross Validation or leave-one-out if $T_{k,t} < 8$).
    Compute the UCB: $B_{k,t} = \hat{r}_{k,t} + \sqrt{\frac{a \log N}{T_{k,t-1}}}$ for each action $1 \leq k \leq K$.
    Select the image to present: $k_t = \arg\max_{k \in \{1,\ldots,K\}} B_{k,t}$.
    Update $T$: $T_{k_t,t} = T_{k_t,t-1} + 1$ and $\forall k \neq k_t, T_{k,t} = T_{k,t-1}$
**end for**

---

Table 1: Pseudo-code of the *UCB-classif* algorithm.

gather as much data as possible from the best action in order to build the best possible classifier. The *UCB-classif* algorithm guarantees that the non-optimal tasks are chosen only a negligible fraction of times ($O(\log N)$ times out of a total budget $N$). The best action is thus sampled $N - O(\log N)$ times (this is formally proven in the Supplementary Material)[3]. It is a huge gain when compared to actual unadaptive procedures for building training sets. Indeed, the unadaptive optimal strategy is to sample each action $N/K$ times, and thus the best task is only sampled $N/K$ times (and not $N - O(\log N)$). More precisely, we prove the following Theorem.

**Theorem 1** *For any $N \geq 2qK$, with probability at least $1 - \frac{1}{N}$, if Equation 1 is satisfied (e.g. if the data are i.i.d.) and if $a \geq 5(d+1)$ we have that the number of times that the image of the best imaginary movement is displayed by algorithm UCB-classif is such that (where $r^* = \max_k r_k^*$)*

$$T_N^* \geq N - \sum_k 8 \frac{a \log(8NK)}{(r^* - r_k^*)^2} \ .$$

The proof of this Theorem is in the provided Supplementary Material, Appendix A.

### 3.3 Discussion on variants of this algorithm

We stated that our objective, given a fixed budget $N$, is to find as fast as possible the image with highest classification rate, and to train the classifier with as many samples as possible. Depending on the objectives of the practitioner, other possible aims can however be pursued. We briefly describe two other settings, and explain how ideas from the bandit setting can be used to adapt to these different scenarios.

**Best stopping time:** A close, yet different, goal, is to find the best time for stopping the training phase. In this setting, the practitioner's aim is to stop the training phase as soon as the algorithm has built an almost optimal classifier for the user. With ideas very similar to those developed in [16] (and extended for bandit problems in e.g. [17]), we can think of an adaptation of algorithm UCB-classif to this new formulation of the problem. Assume that the objective is to find an $\epsilon-$optimal classifier with probability $1 - \delta$, and to stop the training phase as soon as this classifier is built. Then using ideas similar to those presented in [17], an efficient algorithm will at time $t$ select the action that maximizes $B'_{k,t} = \hat{r}_{k,t} + \sqrt{\frac{a \log(NK/\delta)}{T_{k,t-1}}}$ and will stop at the first time $\hat{T}$ when there is an action $\hat{k}^*$ such that $\forall k \neq \hat{k}^*$, $B'_{\hat{k}^*,\hat{T}} - B'_{k,\hat{T}} > \epsilon + 2\sqrt{\frac{a \log(NK/\delta)}{T_{k,\hat{T}-1}}}$ . We thus shorten the training phase almost optimally on the class of adaptive algorithms (see [17] for more details).

**Choice of the best action with a limited budget:** Another question that could be of interest for the practitioner is to find the best action with a fixed budget (and not train the classifier at the same time). We can use ideas from paper [18] to modify UCB-classif. By selecting at each time $t$ the action that maximizes $B''_{k,t} = \hat{r}_{k,t} + \sqrt{\frac{a(N-K)}{T_{k,t-1}}}$, we attain this objective in the sense that we guarantee that the probability of choosing a non-optimal action decreases exponentially fast with $N$.

## 4 Results

We present some numerical experiments illustrating the efficiency of Bandit algorithms for this problem. Although the objective is to implement UCB-classif on the BCI device, in this paper we test the algorithm on real databases that we bootstrap (this is explained in details later). This kind of

procedure is common for testing the performances of adaptive algorithms (see e.g. [19]). Acquiring data for BCI experiments is time-consuming because it requires a human subject to sit through the experiment. The advantage of bootstrapping is that several experiments can be performed with a single database, making it possible to provide confidence bands for the results.

In this Section, we present the experiments we performed, i.e. describe the kind of data we collect, and illustrate the performance of our algorithm on these data.

## 4.1 Performances of the different tasks

The images that were displayed to the subjects correspond to movements of *both feet*, of the *tongue*, of the *right hand*, and of the *left hand* (4 actions in total). Six right-handed subjects went through the experiment with real movements and three of them went through an additional shorter experiment with imaginary movements. For four of the six subjects, the best performance for the real movement was achieved with the *right hand*, whereas the two other subjects' best tasks corresponded to the *left hand* and the *feet*. We collected data for these four tasks. It is not a large number of tasks but we needed a large amount of data for each of them in order to do a significant comparison. In order to have a larger number of tasks and place ourselves in a more realistic situation, we created some articicial tasks (see below). Results on only four tasks are presented in a companion article [20].

Surprisingly, two of the subjects who went through the imaginary experiment obtained better results while imagining moving their *left hand* than their *right hand*, which was the best task during the real movements experiment. For the third subject who did the imaginary experiment, the best task was the *feet*, as for the real movement experiment.

As explained in section 2.2, for this study we chose to use a very small set of fixed features (12 features, extracted from 3 electrodes, 2 frequency bands and 2 time-windows), calibrated on only one of the six subjects during a preliminary experiment. In this work, the features were not subject-specific. It would certainly improve the classification results to tune the features. Using the bandit algorithm to tune the features and to select the tasks at the same time presents a risk overfitting, especially for an initially very small amount of data, and also a risk of biasing the task selection to those that have been the most sampled, and for which the features will thus be the best tuned. Although for all the subjects, the best task achieved a classification accuracy above $85\%$, this accuracy could further be improved by using a larger set of subject-specific features [21] and more advanced techniques (like the CSP [22] or feature selection [23]).

## 4.2 Performances of the bandit algorithm

We compare the performance of the *UCB-classif* sampling strategy to a *uniform* strategy, i.e. the standard way of selecting a task, consisting of $N/K$ presentations of each image.

| Movement | Number of presentations | Off-line classification rate |
|---|---|---|
| Right hand | $28.6 \pm 12.8$ | 88.1% |
| Left hand | $9.0 \pm 7.5$ | 80.5% |
| Feet | $11.6 \pm 9.5$ | 82.6% |
| Tongue | $4.5 \pm 1.5$ | 63.3% |
| Feet 80% | $5.1 \pm 2.6$ | 71.4% |
| Feet 60% | $4.0 \pm 1.5$ | 68.6% |
| Feet 40% | $3.5 \pm 1.0$ | 59.2% |
| Feet 20% | $3.5 \pm 0.9$ | 54.0% |
| Total presentations | 70 | |

Table 2: Actions presented by the UCB-classif algorithm for subject 5 across 500 simulated online BCI experiments. Feet X% is a mixture of the features measured during feet movement and during the resting condition, with a X/100-X proportion. (The off-line classification rate of each action gives an idea of the performance of each action).

To obtain a realistic evaluation of the performance of our algorithm we use a bootstrap technique. More precisely, for each chosen budget $N$, for the *UCB-classif* strategy and the *uniform* strategy, we simulated 500 online BCI experiments by randomly sampling from the acquired data of each action.

Table 2 shows, for one subject and for a fixed budget of $N = 70$, the average number of presentations of each task $T_k$, and its standard deviation, across the 500 simulated experiments. It also contains the off-line classification rate of each task to give an idea of the performances of the different tasks for this subject. We can see that very little budget is allocated to the *tongue* movement and to the most degraded *feet 20%* tasks, which are the less discriminative actions, and that most of the budget is devoted to the *right hand*, thus enabling a more efficient training.

Figure 2 and Table 3 show, for different budgets ($N$), the performance of the *UCB-classif* algorithm versus the uniform technique. The training of the classifier is done on the actions presented during the simulated BCI experiment, and the testing on the remaining data.

For a budget $N > 70$ the *UCB-classif* could not be used for all the subjects because there was not enough data for the best action (One subject only underwent a session of 5 blocks and so only 50 samples of each motor task were recorded. If we try to simulate an on-line experiment using the *UCB-classif* with a budget higher than $N = 70$ it is likely to ask for a 51th presentation of the best task, which has not been recorded).

The classification results depend on which data is used to simulate the BCI experiment. To give an idea of this variability, the first and last quartiles are plotted as error bars on the graphics.

| Budget ($N$) | Length of the experiment | Uniform strategy | *UCB-classif* | Benefit |
|---|---|---|---|---|
| 30 | 3min45 | 47.7% | 64.4% | +16.7% |
| 40 | 5min | 58.5% | 77.2% | +18.7% |
| 50 | 6min15 | 63.4% | 82.0% | +18.5% |
| 60 | 7min30 | 67.0% | 84.0% | +17.1% |
| 70 | 8min45 | 70.1% | 85.7% | +15.6% |
| 100 | 12min30 | 77.6% | * | |
| 150 | 18min45 | 83.2% | * | |
| 180 | 22min30 | 85.2% | * | |

Table 3: Comparison of the performances of the *UCB-classif* vs. the *uniform* strategy for different budgets, averaged over all subjects, for real movements. (The increases are significant with $p > 95\%$.) For each budget, we give an indication of the length of the experiment (without counting pauses between blocks) required to obtain this amount of data.

The *UCB-classif* strategy significantly outperforms the *uniform* strategy, even for relatively small $N$. On average on all the users it even gives better classification rates when using only half of the available samples, compared to the uniform strategy. Indeed, Table 3 shows that, to achieve a classification rate of 85% the *UCB-classif* only requires a budget of $N = 70$ whereas the uniform strategy needs $N = 180$. We believe that such gain in performance motivates the implementation of such a training algorithm in BCI devices, specially since the algorithm itself is quite simple and fast.

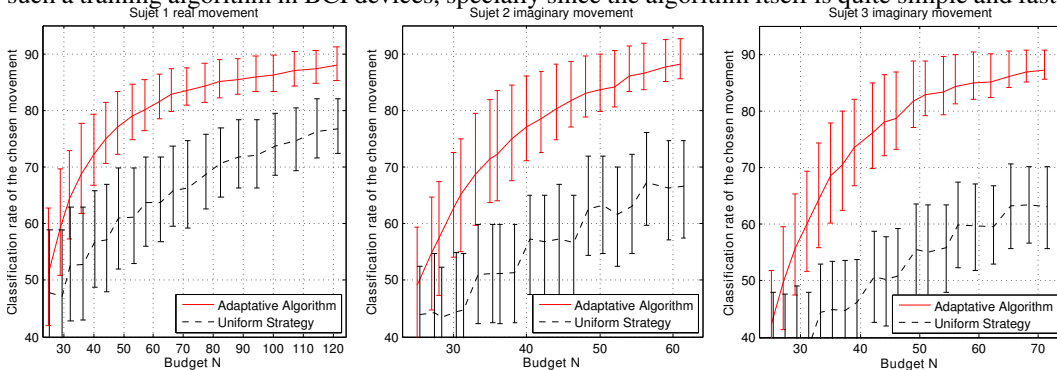

Figure 2: *UCB-classif* algorithm (full line, red) versus uniform strategy (dashed line, black).

## 5 Conclusion

The method presented in this paper falls in the category of adaptive BCI based on Bandit Theory. To the best of our knowledge, this is the first such method for dealing with automatic task selection. *UCB-classif* is a new adaptive algorithm that allows to automatically select a motor task in view of a brain-controlled button. By rapidly eliminating non-efficient motor tasks and focusing on the most promising ones, it enables a better task selection procedure than a uniform strategy. Moreover, by more frequently presenting the best task it allows a good training of the classifier. This algorithm enables to shorten the training period, or equivalently, to allow for a larger set of possible movements among which to select the best. In a paper due to appear [20], we implement this algorithm online. A future research direction is to learn several discriminant tasks in order to activate several buttons.

**Acknowledgements** This work was partially supported by the French ANR grant Co-Adapt ANR-09-EMER-002, Nord-Pas-de-Calais Regional Council, French ANR grant EXPLO-RA (ANR-08-COSI-004), the EC Seventh Framework Programme (FP7/2007-2013) under grant agreement 270327 (CompLACS project), and by Pascal-2.

## Footnotes

[1]Here, the actions are images displayed to the BCI user as prompts to perform the corresponding imaginary tasks.

[2]The tasks correspond to the imaginary movements of moving the *feet*, *tongue*, *right hand*, and *left hand*, plus 4 additional degraded tasks (so a total of $K = 8$ actions).

[3]The ideas of the proof are very similar to the ideas in [10], with the difference that the upper bounds have to be computed using inequalities based on VC-dimension.

# References

[1] D. J. McFarland, W. A. Sarnacki, and J. R Wolpaw. Electroencephalographic (EEG) control of three-dimensional movement. *Journal of Neural Engineering*, 7(3):036007, 2010.

[2] T. Solis-Escalante, G. Mller-Putz, C. Brunner, V. Kaiser, and G. Pfurtscheller. Analysis of sensorimotor rhythms for the implementation of a brain switch for healthy subjects. *Biomedical Signal Processing and Control*, 5(1):15 – 20, 2010.

[3] B. Blankertz, G. Dornhege, M. Krauledat, K.-R. Mller, and G. Curio. The non-invasive berlin brain-computer interface: Fast acquisition of effective performance in untrained subjects. *NeuroImage*, 37(2):539 – 550, 2007.

[4] J. Fruitet, M. Clerc, and T. Papadopoulo. Preliminary study for an hybrid BCI using sensorimotor rhythms and beta rebound. In *International Journal of Bioelectromagnetism*, 2011.

[5] J. R. Wolpaw, N. Birbaumer, D. J. McFarland, G. Pfurtscheller, and T. M. Vaughan. Brain-computer interfaces for communication and control. *Clinical Neurophysiology*, 113(6):767 – 791, 2002.

[6] J. del R. Millán, F. Renkens, J. Mourio, and W. Gerstner. Brain-actuated interaction. *Artificial Intelligence*, 159(1-2):241 – 259, 2004.

[7] C. Vidaurre and B. Blankertz. Towards a cure for BCI illiteracy. *Brain Topography*, 23:194–198, 2010. 10.1007/s10548-009-0121-6.

[8] M.-C. Dobrea and D.M. Dobrea. The selection of proper discriminative cognitive tasks - a necessary prerequisite in high-quality BCI applications. In *Applied Sciences in Biomedical and Communication Technologies, 2009. ISABEL 2009. 2nd International Symposium on*, pages 1 –6, 2009.

[9] H. Robbins. Some aspects of the sequential design of experiments. *Bulletin of the American Mathematics Society*, 58:527–535, 1952.

[10] P. Auer, N. Cesa-Bianchi, and P. Fischer. Finite time analysis of the multiarmed bandit problem. *Machine Learning*, 47(2-3):235–256, 2002.

[11] G. Pfurtscheller and F. H. Lopes da Silva. Event-related EEG/MEG synchronization and desynchronization: basic principles. *Clinical Neurophysiology*, 110(11):1842 – 1857, 1999.

[12] G. Pfurtscheller and C. Neuper. Motor imagery activates primary sensorimotor area in humans. *Neuroscience Letters*, 239(2-3):65 – 68, 1997.

[13] H. Jasper and W. Penfield. Electrocorticograms in man: Effect of voluntary movement upon the electrical activity of the precentral gyrus. *European Archives of Psychiatry and Clinical Neuroscience*, 183:163–174, 1949. 10.1007/BF01062488.

[14] Y. Renard, F. Lotte, G. Gibert, M. Congedo, E. Maby, V. Delannoy, O. Bertrand, and A. Lécuyer. OpenViBE: An open-source software platform to design, test, and use brain–computer interfaces in real and virtual environments. *Presence: Teleoperators and Virtual Environments*, 19(1):35–53, 2010.

[15] V.N. Vapnik. *The nature of statistical learning theory*. Springer-Verlag New York Inc, 2000.

[16] O. Maron and A.W. Moore. Hoeffding races: Accelerating model selection search for classification and function approximation. *Robotics Institute*, page 263, 1993.

[17] J.Y. Audibert, S. Bubeck, and R. Munos. Bandit view on noisy optimization. *Optimization for Machine Learning*, pages 431–454, 2011.

[18] J.Y. Audibert, S. Bubeck, and R. Munos. Best arm identification in multi-armed bandits. In *Annual Conference on Learning Theory (COLT)*, 2010.

[19] J. Langford, A. Strehl, and J. Wortman. Exploration scavenging. In *Proceedings of the 25th international conference on Machine learning*, pages 528–535. ACM, 2008.

[20] J. Fruitet, A. Carpentier, R. Munos, M. Clerc, et al. Automatic motor task selection via a bandit algorithm for a brain-controlled button. *Journal of Neural Engineering*, 2012. (to appear).

[21] M. Dobrea, D.M. Dobrea, and D. Alexa. Spectral EEG features and tasks selection process: Some considerations toward BCI applications. In *Multimedia Signal Processing (MMSP), 2010 IEEE International Workshop on*, pages 150 –155, 2010.

[22] H. Ramoser, J. Muller-Gerking, and G. Pfurtscheller. Optimal spatial filtering of single trial EEG during imagined hand movement. *Rehabilitation Engineering, IEEE Transactions on*, 8(4):441 –446, 2000.

[23] J. Fruitet, D. J. McFarland, and J. R. Wolpaw. A comparison of regression techniques for a two-dimensional sensorimotor rhythm-based brain-computer interface. *Journal of Neural Engineering*, 7(1), 2010.

